# Two-Dimensional Linear Discriminant Analysis

**Jieping Ye**
Department of CSE
University of Minnesota
jieping@cs.umn.edu

**Ravi Janardan**
Department of CSE
University of Minnesota
janardan@cs.umn.edu

**Qi Li**
Department of CIS
University of Delaware
qili@cis.udel.edu

## Abstract

Linear Discriminant Analysis (LDA) is a well-known scheme for feature extraction and dimension reduction. It has been used widely in many applications involving high-dimensional data, such as face recognition and image retrieval. An intrinsic limitation of classical LDA is the so-called *singularity problem*, that is, it fails when all scatter matrices are singular. A well-known approach to deal with the singularity problem is to apply an intermediate dimension reduction stage using Principal Component Analysis (PCA) before LDA. The algorithm, called PCA+LDA, is used widely in face recognition. However, PCA+LDA has high costs in time and space, due to the need for an eigen-decomposition involving the scatter matrices.

In this paper, we propose a novel LDA algorithm, namely 2DLDA, which stands for **2-D**imensional **L**inear **D**iscriminant **A**nalysis. 2DLDA overcomes the singularity problem implicitly, while achieving efficiency. The key difference between 2DLDA and classical LDA lies in the model for data representation. Classical LDA works with vectorized representations of data, while the 2DLDA algorithm works with data in matrix representation. To further reduce the dimension by 2DLDA, the combination of 2DLDA and classical LDA, namely 2DLDA+LDA, is studied, where LDA is preceded by 2DLDA. The proposed algorithms are applied on face recognition and compared with PCA+LDA. Experiments show that 2DLDA and 2DLDA+LDA achieve competitive recognition accuracy, while being much more efficient.

## 1   Introduction

Linear Discriminant Analysis [2, 4] is a well-known scheme for feature extraction and dimension reduction. It has been used widely in many applications such as face recognition [1], image retrieval [6], microarray data classification [3], etc. Classical LDA projects the data onto a lower-dimensional vector space such that the ratio of the between-class distance to the within-class distance is maximized, thus achieving maximum discrimination. The optimal projection (transformation) can be readily computed by applying the eigen-decomposition on the scatter matrices. An intrinsic limitation of classical LDA is that its objective function requires the nonsingularity of one of the scatter matrices. For many applications, such as face recognition, all scatter matrices in question can be singular since the data is from a very high-dimensional space, and in general, the dimension exceeds the

number of data points. This is known as the *undersampled* or *singularity* problem [5].

In recent years, many approaches have been brought to bear on such high-dimensional, undersampled problems, including pseudo-inverse LDA, PCA+LDA, and regularized LDA. More details can be found in [5]. Among these LDA extensions, PCA+LDA has received a lot of attention, especially for face recognition [1]. In this two-stage algorithm, an intermediate dimension reduction stage using PCA is applied before LDA. The common aspect of previous LDA extensions is the computation of eigen-decomposition of certain large matrices, which not only degrades the efficiency but also makes it hard to scale them to large datasets.

In this paper, we present a novel approach to alleviate the expensive computation of the eigen-decomposition in previous LDA extensions. The novelty lies in a different data representation model. Under this model, each datum is represented as a matrix, instead of as a vector, and the collection of data is represented as a collection of matrices, instead of as a single large matrix. This model has been previously used in [8, 9, 7] for the generalization of SVD and PCA. Unlike classical LDA, we consider the projection of the data onto a space which is the tensor product of two vector spaces. We formulate our dimension reduction problem as an optimization problem in Section 3. Unlike classical LDA, there is no closed form solution for the optimization problem; instead, we derive a heuristic, namely 2DLDA. To further reduce the dimension, which is desirable for efficient querying, we consider the combination of 2DLDA and LDA, namely 2DLDA+LDA, where the dimension of the space transformed by 2DLDA is further reduced by LDA.

We perform experiments on three well-known face datasets to evaluate the effectiveness of 2DLDA and 2DLDA+LDA and compare with PCA+LDA, which is used widely in face recognition. Our experiments show that: (1) 2DLDA is applicable to high-dimensional undersampled data such as face images, i.e., it implicitly avoids the singularity problem encountered in classical LDA; and (2) 2DLDA and 2DLDA+LDA have distinctly lower costs in time and space than PCA+LDA, and achieve classification accuracy that is competitive with PCA+LDA.

## 2   An overview of LDA

In this section, we give a brief overview of classical LDA. Some of the important notations used in the rest of this paper are listed in Table 1.

Given a data matrix $A \in \mathbb{R}^{N \times n}$, classical LDA aims to find a transformation $G \in \mathbb{R}^{N \times \ell}$ that maps each column $a_i$ of $A$, for $1 \leq i \leq n$, in the $N$-dimensional space to a vector $b_i$ in the $\ell$-dimensional space. That is $G : a_i \in \mathbb{R}^N \to b_i = G^T a_i \in \mathbb{R}^\ell (\ell < N)$. Equivalently, classical LDA aims to find a vector space $\mathcal{G}$ spanned by $\{g_i\}_{i=1}^\ell$, where $G = [g_1, \cdots, g_\ell]$, such that each $a_i$ is projected onto $\mathcal{G}$ by $(g_1^T \cdot a_i, \cdots, g_\ell^T \cdot a_i)^T \in \mathbb{R}^\ell$.

Assume that the original data in $A$ is partitioned into $k$ classes as $A = \{\Pi_1, \cdots, \Pi_k\}$, where $\Pi_i$ contains $n_i$ data points from the $i$th class, and $\sum_{i=1}^k n_i = n$. Classical LDA aims to find the optimal transformation $G$ such that the class structure of the original high-dimensional space is preserved in the low-dimensional space.

In general, if each class is tightly grouped, but well separated from the other classes, the quality of the cluster is considered to be high. In discriminant analysis, two scatter matrices, called *within-class* ($S_w$) and *between-class* ($S_b$) matrices, are defined to quantify the quality of the cluster, as follows [4]: $S_w = \sum_{i=1}^k \sum_{x \in \Pi_i} (x - m_i)(x - m_i)^T$, and $S_b = \sum_{i=1}^k n_i (m_i - m)(m_i - m)^T$, where $m_i = \frac{1}{n_i} \sum_{x \in \Pi_i} x$ is the *mean* of the $i$th class, and $m = \frac{1}{n} \sum_{i=1}^k \sum_{x \in \Pi_i} x$ is the *global mean*.

| Notation | Description |
|---|---|
| $n$ | number of images in the dataset |
| $k$ | number of classes in the dataset |
| $A_i$ | $i$th image in matrix representation |
| $a_i$ | $i$th image in vectorized representation |
| $r$ | number of rows in $A_i$ |
| $c$ | number of columns in $A_i$ |
| $N$ | dimension of $a_i$ ($N = r * c$) |
| $\Pi_j$ | $j$th class in the dataset |
| $L$ | transformation matrix (left) by 2DLDA |
| $R$ | transformation matrix (right) by 2DLDA |
| $I$ | number of iterations in 2DLDA |
| $B_i$ | reduced representation of $A_i$ by 2DLDA |
| $\ell_1$ | number of rows in $B_i$ |
| $\ell_2$ | number of columns in $B_i$ |

Table 1: Notation

It is easy to verify that trace($S_w$) measures the closeness of the vectors within the classes, while trace($S_b$) measures the separation between classes. In the low-dimensional space resulting from the linear transformation $G$ (or the linear projection onto the vector space $\mathcal{G}$), the within-class and between-class matrices become $S_b^L = G^T S_b G$, and $S_w^L = G^T S_w G$.

An optimal transformation $G$ would maximize trace($S_b^L$) and minimize trace($S_w^L$). Common optimizations in classical discriminant analysis include (see [4]):

$$\max_G \left\{ \text{trace}((S_w^L)^{-1} S_b^L) \right\} \text{ and } \min_G \left\{ \text{trace}((S_b^L)^{-1} S_w^L) \right\}. \tag{1}$$

The optimization problems in Eq. (1) are equivalent to the following generalized eigenvalue problem: $S_b x = \lambda S_w x$, for $\lambda \neq 0$. The solution can be obtained by applying an eigen-decomposition to the matrix $S_w^{-1} S_b$, if $S_w$ is nonsingular, or $S_b^{-1} S_w$, if $S_b$ is nonsingular. There are at most $k-1$ eigenvectors corresponding to nonzero eigenvalues, since the rank of the matrix $S_b$ is bounded from above by $k-1$. Therefore, the reduced dimension by classical LDA is at most $k-1$. A stable way to compute the eigen-decomposition is to apply SVD on the scatter matrices. Details can be found in [6].

Note that a limitation of classical LDA in many applications involving undersampled data, such as text documents and images, is that at least one of the scatter matrices is required to be nonsingular. Several extensions, including pseudo-inverse LDA, regularized LDA, and PCA+LDA, were proposed in the past to deal with the singularity problem. Details can be found in [5].

## 3   2-Dimensional LDA

The key difference between classical LDA and the 2DLDA that we propose in this paper is in the representation of data. While classical LDA uses the vectorized representation, 2DLDA works with data in matrix representation.

We will see later in this section that the matrix representation in 2DLDA leads to an eigen-decomposition on matrices with much smaller sizes. More specifically, 2DLDA involves the eigen-decomposition of matrices with sizes $r \times r$ and $c \times c$, which are much smaller than the matrices in classical LDA. This dramatically reduces the time and space complexities of 2DLDA over LDA.

Unlike classical LDA, 2DLDA considers the following $(\ell_1 \times \ell_2)$-dimensional space $\mathcal{L} \otimes \mathcal{R}$, which is the tensor product of the following two spaces: $\mathcal{L}$ spanned by $\{u_i\}_{i=1}^{\ell_1}$ and $\mathcal{R}$ spanned by $\{v_i\}_{i=1}^{\ell_2}$. Define two matrices $L = [u_1, \cdots, u_{\ell_1}] \in \mathbb{R}^{r \times \ell_1}$ and $R = [v_1, \cdots, v_{\ell_2}] \in \mathbb{R}^{c \times \ell_2}$. Then the projection of $X \in \mathbb{R}^{r \times c}$ onto the space $\mathcal{L} \otimes \mathcal{R}$ is $L^T X R \in \mathbb{R}^{\ell_1 \times \ell_2}$.

Let $A_i \in \mathbb{R}^{r \times c}$, for $i = 1, \cdots, n$, be the $n$ images in the dataset, clustered into classes $\Pi_1, \cdots, \Pi_k$, where $\Pi_i$ has $n_i$ images. Let $M_i = \frac{1}{n_i} \sum_{X \in \Pi_i} X$ be the mean of the $i$th class, $1 \leq i \leq k$, and $M = \frac{1}{n} \sum_{i=1}^{k} \sum_{X \in \Pi_i} X$ be the global mean. In 2DLDA, we consider images as two-dimensional signals and aim to find two transformation matrices $L \in \mathbb{R}^{r \times \ell_1}$ and $R \in \mathbb{R}^{c \times \ell_2}$ that map each $A_i \in \mathbb{R}^{r \times c}$, for $1 \leq i \leq n$, to a matrix $B_i \in \mathbb{R}^{\ell_1 \times \ell_2}$ such that $B_i = L^T A_i R$.

Like classical LDA, 2DLDA aims to find the optimal transformations (projections) $L$ and $R$ such that the class structure of the original high-dimensional space is preserved in the low-dimensional space.

A natural similarity metric between matrices is the Frobenius norm [8]. Under this metric, the (squared) within-class and between-class distances $D_w$ and $D_b$ can be computed as follows:

$$D_w = \sum_{i=1}^{k} \sum_{X \in \Pi_i} ||X - M_i||_F^2, \quad D_b = \sum_{i=1}^{k} n_i ||M_i - M||_F^2.$$

Using the property of the *trace*, that is, $\text{trace}(MM^T) = ||M||_F^2$, for any matrix $M$, we can rewrite $D_w$ and $D_b$ as follows:

$$D_w = \text{trace}\left( \sum_{i=1}^{k} \sum_{X \in \Pi_i} (X - M_i)(X - M_i)^T \right),$$

$$D_b = \text{trace}\left( \sum_{i=1}^{k} n_i (M_i - M)(M_i - M)^T \right).$$

In the low-dimensional space resulting from the linear transformations $L$ and $R$, the within-class and between-class distances become

$$\tilde{D}_w = \text{trace}\left( \sum_{i=1}^{k} \sum_{X \in \Pi_i} L^T (X - M_i) R R^T (X - M_i)^T L \right),$$

$$\tilde{D}_b = \text{trace}\left( \sum_{i=1}^{k} n_i L^T (M_i - M) R R^T (M_i - M)^T L \right).$$

The optimal transformations $L$ and $R$ would maximize $\tilde{D}_b$ and minimize $\tilde{D}_w$. Due to the difficulty of computing the optimal $L$ and $R$ simultaneously, we derive an iterative algorithm in the following. More specifically, for a fixed $R$, we can compute the optimal $L$ by solving an optimization problem similar to the one in Eq. (1). With the computed $L$, we can then update $R$ by solving another optimization problem as the one in Eq. (1). Details are given below. The procedure is repeated a certain number of times, as discussed in Section 4.

**Computation of $L$**

For a fixed $R$, $\tilde{D}_w$ and $\tilde{D}_b$ can be rewritten as
$$\tilde{D}_w = \text{trace}\left( L^T S_w^R L \right), \quad \tilde{D}_b = \text{trace}\left( L^T S_b^R L \right),$$

---

**Algorithm 2DLDA**$(A_1, \cdots, A_n, \ell_1, \ell_2)$
**Input:**   $A_1, \cdots, A_n, \ell_1, \ell_2$
**Output:**   $L, R, B_1, \cdots, B_n$

1. Compute the mean $M_i$ of $i$th class for each $i$ as $M_i = \frac{1}{n_i} \sum_{X \in \Pi_i} X$;

2. Compute the global mean $M = \frac{1}{n} \sum_{i=1}^{k} \sum_{X \in \Pi_i} X$;

3. $R_0 \leftarrow (I_{\ell_2}, 0)^T$;

4. For $j$ from 1 to I

5.    $S_w^R \leftarrow \sum_{i=1}^{k} \sum_{X \in \Pi_i} (X - M_i) R_{j-1} R_{j-1}^T (X - M_i)^T$,
   $S_b^R \leftarrow \sum_{i=1}^{k} n_i (M_i - M) R_{j-1} R_{j-1}^T (M_i - M)^T$;

6.    Compute the first $\ell_1$ eigenvectors $\{\phi_\ell^L\}_{\ell=1}^{\ell_1}$ of $(S_w^R)^{-1} S_b^R$;

7.    $L_j \leftarrow [\phi_1^L, \cdots, \phi_{\ell_1}^L]$

8.    $S_w^L \leftarrow \sum_{i=1}^{k} \sum_{X \in \Pi_i} (X - M_i)^T L_j L_j^T (X - M_i)$,
   $S_b^L \leftarrow \sum_{i=1}^{k} n_i (M_i - M)^T L_j L_j^T (M_i - M)$;

9.    Compute the first $\ell_2$ eigenvectors $\{\phi_\ell^R\}_{\ell=1}^{\ell_2}$ of $(S_w^L)^{-1} S_b^L$;

10.   $R_j \leftarrow [\phi_1^R, \cdots, \phi_{\ell_2}^R]$;

11. EndFor

12. $L \leftarrow L_I, R \leftarrow R_I$;

13. $B_\ell \leftarrow L^T A_\ell R$, for $\ell = 1, \cdots, n$;

14. return$(L, R, B_1, \cdots, B_n)$.

---

where

$$S_w^R = \sum_{i=1}^{k} \sum_{X \in \Pi_i} (X - M_i) R R^T (X - M_i)^T, \quad S_b^R = \sum_{i=1}^{k} n_i (M_i - M) R R^T (M_i - M)^T.$$

Similar to the optimization problem in Eq. (1), the optimal $L$ can be computed by solving the following optimization problem: $\max_L \text{trace} \left((L^T S_w^R L)^{-1}(L^T S_b^R L)\right)$. The solution can be obtained by solving the following generalized eigenvalue problem: $S_w^R x = \lambda S_b^R x$. Since $S_w^R$ is in general nonsingular, the optimal $L$ can be obtained by computing an eigen-decomposition on $(S_w^R)^{-1} S_b^R$. Note that the size of the matrices $S_w^R$ and $S_b^R$ is $r \times r$, which is much smaller than the size of the matrices $S_w$ and $S_b$ in classical LDA.

**Computation of $R$**

Next, consider the computation of $R$, for a fixed $L$. A key observation is that $\tilde{D}_w$ and $\tilde{D}_b$ can be rewritten as

$$\tilde{D}_w = \text{trace} \left(R^T S_w^L R\right), \quad \tilde{D}_b = \text{trace} \left(R^T S_b^L R\right),$$

where

$$S_w^L = \sum_{i=1}^{k} \sum_{X \in \Pi_i} (X - M_i)^T L L^T (X - M_i), \quad S_b^L = \sum_{i=1}^{k} n_i (M_i - M)^T L L^T (M_i - M).$$

This follows from the following property of trace, that is, $\text{trace}(AB) = \text{trace}(BA)$, for any two matrices $A$ and $B$.

Similarly, the optimal $R$ can be computed by solving the following optimization problem: $\max_R \text{trace} \left((R^T S_w^L R)^{-1}(R^T S_b^L R)\right)$. The solution can be obtained by solving the following generalized eigenvalue problem: $S_w^L x = \lambda S_b^L x$. Since $S_w^L$ is in general nonsingular,

the optimal $R$ can be obtained by computing an eigen-decomposition on $\left(S_w^L\right)^{-1} S_b^L$. Note that the size of the matrices $S_w^L$ and $S_b^L$ is $c \times c$, much smaller than $S_w$ and $S_b$.

The pseudo-code for the 2DLDA algorithm is given in **Algorithm 2DLDA**. It is clear that the most expensive steps in **Algorithm 2DLDA** are in Lines 5, 8 and 13 and the total time complexity is $O(n \max(\ell_1, \ell_2)(r + c)^2 I)$, where $I$ is the number of iterations. The 2DLDA algorithm depends on the initial choice $R_0$. Our experiments show that choosing $R_0 = (I_{\ell_2}, 0)^T$, where $I_{\ell_2}$ is the identity matrix, produces excellent results. We use this initial $R_0$ in all the experiments.

Since the number of rows ($r$) and the number of columns ($c$) of an image $A_i$ are generally comparable, i.e., $r \approx c \approx \sqrt{N}$, we set $\ell_1$ and $\ell_2$ to a common value $d$ in the rest of this paper, for simplicity. However, the algorithm works in the general case. With this simplification, the time complexity of the 2DLDA algorithm becomes $O(ndNI)$.

The space complexity of 2DLDA is $O(rc) = O(N)$. The key to the low space complexity of the algorithm is that the matrices $S_w^R$, $S_b^R$, $S_w^L$, and $S_b^L$ can be formed by reading the matrices $A_\ell$ incrementally.

### 3.1   2DLDA+LDA

As mentioned in the Introduction, PCA is commonly applied as an intermediate dimension-reduction stage before LDA to overcome the singularity problem of classical LDA. In this section, we consider the combination of 2DLDA and LDA, namely 2DLDA+LDA, where the dimension by 2DLDA is further reduced by LDA, since small reduced dimension is desirable for efficient querying. More specifically, in the first stage of 2DLDA+LDA, each image $A_i \in \mathbb{R}^{r \times c}$ is reduced to $B_i \in \mathbb{R}^{d \times d}$ by 2DLDA, with $d < \min(r, c)$. In the second stage, each $B_i$ is first transformed to a vector $b_i \in \mathbb{R}^{d^2}$ (*matrix-to-vector alignment*), then $b_i$ is further reduced to $b_i^L \in \mathbb{R}^{k-1}$ by LDA with $k - 1 < d^2$, where $k$ is the number of classes. Here, "matrix-to-vector alignment" means that the matrix is transformed to a vector by concatenating all its rows together consecutively.

The time complexity of the first stage by 2DLDA is $O(ndNI)$. The second stage applies classical LDA to data in $d^2$-dimensional space, hence takes $O(n(d^2)^2)$, assuming $n > d^2$. Hence the total time complexity of 2DLDA+LDA is $O\left(nd(NI + d^3)\right)$.

## 4   Experiments

In this section, we experimentally evaluate the performance of 2DLDA and 2DLDA+LDA on face images and compare with PCA+LDA, used widely in face recognition. For PCA+LDA, we use 200 principal components in the PCA stage, as it produces good overall results. All of our experiments are performed on a P4 1.80GHz Linux machine with 1GB memory. For all the experiments, the 1-Nearest-Neighbor (1NN) algorithm is applied for classification and ten-fold cross validation is used for computing the classification accuracy.

**Datasets:**   We use three face datasets in our study: PIX, ORL, and PIE, which are publicly available. PIX (available at http://peipa.essex.ac.uk/ipa/pix/faces/manchester/test-hard/), contains 300 face images of 30 persons. The image size is $512 \times 512$. We subsample the images down to a size of $100 \times 100 = 10000$. ORL (available at http://www.uk.research.att.com/facedatabase.html), contains 400 face images of 40 persons. The image size is $92 \times 112$. PIE is a subset of the CMU–PIE face image dataset (available at http://www.ri.cmu.edu/projects/project_418.html). It contains 6615 face images of 63 persons. The image size is $640 \times 480$. We subsample the images down to a size of $220 \times 175 = 38500$. Note that PIE is much larger than the other two datasets.

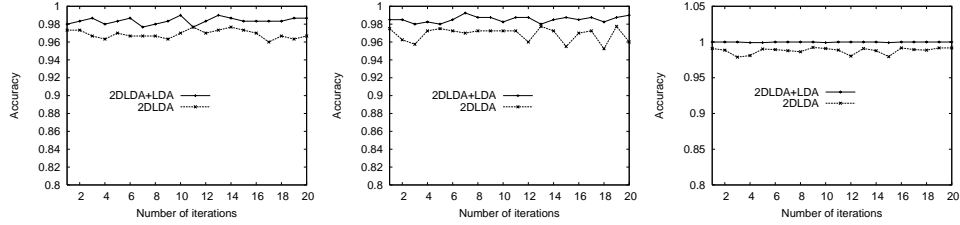

Figure 1: Effect of the number of iterations on 2DLDA and 2DLDA+LDA using the three face datasets; PIX, ORL and PIE (from left to right).

**The impact of the number, $I$, of iterations:** In this experiment, we study the effect of the number of iterations ($I$ in **Algorithm 2DLDA**) on 2DLDA and 2DLDA+LDA. The results are shown in Figure 1, where the $x$-axis denotes the number of iterations, and the $y$-axis denotes the classification accuracy. $d = 10$ is used for both algorithms. It is clear that both accuracy curves are stable with respect to the number of iterations. In general, the accuracy curves of 2DLDA+LDA are slightly more stable than those of 2DLDA. The key consequence is that we only need to run the "for" loop (from Line 4 to Line 11) in **Algorithm 2DLDA** only once, i.e., $I = 1$, which significantly reduces the total running time of both algorithms.

**The impact of the value of the reduced dimension $d$:** In this experiment, we study the effect of the value of $d$ on 2DLDA and 2DLDA+LDA, where the value of $d$ determines the dimensionality in the transformed space by 2DLDA. We did extensive experiments using different values of $d$ on the face image datasets. The results are summarized in Figure 2, where the $x$-axis denotes the values of $d$ (between 1 and 15) and the $y$-axis denotes the classification accuracy with 1-Nearest-Neighbor as the classifier. As shown in Figure 2, the accuracy curves on all datasets stabilize around $d = 4$ to 6.

**Comparison on classification accuracy and efficiency:** In this experiment, we evaluate the effectiveness of the proposed algorithms in terms of classification accuracy and efficiency and compare with PCA+LDA. The results are summarized in Table 2. We can observe that 2DLDA+LDA has similar performance as PCA+LDA in classification, while it outperforms 2DLDA. Hence the LDA stage in 2DLDA+LDA not only reduces the dimension, but also increases the accuracy. Another key observation from Table 2 is that 2DLDA is almost one order of magnitude faster than PCA+LDA, while, the running time of 2DLDA+LDA is close to that of 2DLDA.

Hence 2DLDA+LDA is a more effective dimension reduction algorithm than PCA+LDA, as it is competitive to PCA+LDA in classification and has the same number of reduced dimensions in the transformed space, while it has much lower time and space costs.

## 5  Conclusions

An efficient algorithm, namely 2DLDA, is presented for dimension reduction. 2DLDA is an extension of LDA. The key difference between 2DLDA and LDA is that 2DLDA works on the matrix representation of images directly, while LDA uses a vector representation. 2DLDA has asymptotically minimum memory requirements, and lower time complexity than LDA, which is desirable for large face datasets, while it implicitly avoids the singularity problem encountered in classical LDA. We also study the combination of 2DLDA and LDA, namely 2DLDA+LDA, where the dimension by 2DLDA is further reduced by LDA. Experiments show that 2DLDA and 2DLDA+LDA are competitive with PCA+LDA, in terms of classification accuracy, while they have significantly lower time and space costs.

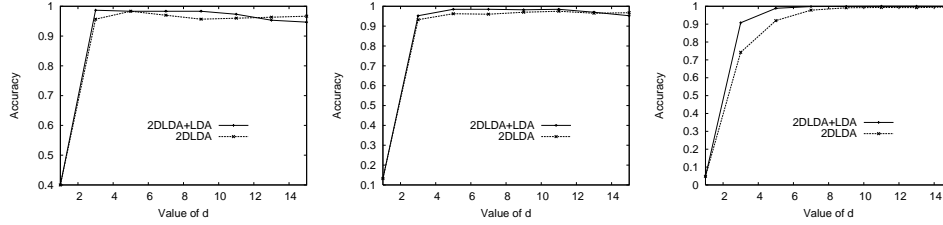

Figure 2: Effect of the value of the reduced dimension $d$ on 2DLDA and 2DLDA+LDA using the three face datasets; PIX, ORL and PIE (from left to right).

| Dataset | PCA+LDA | | 2DLDA | | 2DLDA+LDA | |
|---------|----------|-----------|----------|-----------|----------|-----------|
| | Accuracy | Time(Sec) | Accuracy | Time(Sec) | Accuracy | Time(Sec) |
| PIX | 98.00% | 7.73 | 97.33% | 1.69 | 98.50% | 1.73 |
| ORL | 97.75% | 12.5 | 97.50% | 2.14 | 98.00% | 2.19 |
| PIE | — | — | 99.32% | 153 | 100% | 157 |

Table 2: Comparison on classification accuracy and efficiency: "—" means that PCA+LDA is not applicable for PIE, due to its large size. Note that PCA+LDA involves an eigen-decomposition of the scatter matrices, which requires the whole data matrix to reside in main memory.

**Acknowledgment**  Research of J. Ye and R. Janardan is sponsored, in part, by the Army High Performance Computing Research Center under the auspices of the Department of the Army, Army Research Laboratory cooperative agreement number DAAD19-01-2-0014, the content of which does not necessarily reflect the position or the policy of the government, and no official endorsement should be inferred.

# References

[1] P.N. Belhumeour, J.P. Hespanha, and D.J. Kriegman. Eigenfaces vs. Fisherfaces: Recognition using class specific linear projection. *IEEE Transactions on Pattern Analysis and Machine Intelligence*, 19(7):711–720, 1997.

[2] R.O. Duda, P.E. Hart, and D. Stork. *Pattern Classification*. Wiley, 2000.

[3] S. Dudoit, J. Fridlyand, and T. P. Speed. Comparison of discrimination methods for the classification of tumors using gene expression data. *Journal of the American Statistical Association*, 97(457):77–87, 2002.

[4] K. Fukunaga. *Introduction to Statistical Pattern Classification*. Academic Press, San Diego, California, USA, 1990.

[5] W.J. Krzanowski, P. Jonathan, W.V McCarthy, and M.R. Thomas. Discriminant analysis with singular covariance matrices: methods and applications to spectroscopic data. *Applied Statistics*, 44:101–115, 1995.

[6] Daniel L. Swets and Juyang Weng. Using discriminant eigenfeatures for image retrieval. *IEEE Transactions on Pattern Analysis and Machine Intelligence*, 18(8):831–836, 1996.

[7] J. Yang, D. Zhang, A.F. Frangi, and J.Y. Yang. Two-dimensional PCA: a new approach to appearance-based face representation and recognition. *IEEE Transactions on Pattern Analysis and Machine Intelligence*, 26(1):131–137, 2004.

[8] J. Ye. Generalized low rank approximations of matrices. In *ICML Conference Proceedings*, pages 887–894, 2004.

[9] J. Ye, R. Janardan, and Q. Li. GPCA: An efficient dimension reduction scheme for image compression and retrieval. In *ACM SIGKDD Conference Proceedings*, pages 354–363, 2004.